# Yggdrasil: An Optimized System for Training Deep Decision Trees at Scale

**Firas Abuzaid**[1], **Joseph Bradley**[2], **Feynman Liang**[3], **Andrew Feng**[4], **Lee Yang**[4],
**Matei Zaharia**[1], **Ameet Talwalkar**[5]
[1]MIT CSAIL, [2]Databricks, [3]University of Cambridge, [4]Yahoo, [5]UCLA

## Abstract

Deep distributed decision trees and tree ensembles have grown in importance due to the need to model increasingly large datasets. However, PLANET, the standard distributed tree learning algorithm implemented in systems such as XGBOOST and Spark MLLIB, scales poorly as data dimensionality and tree depths grow. We present YGGDRASIL, a new distributed tree learning method that outperforms existing methods by up to 24×. Unlike PLANET, YGGDRASIL is based on vertical partitioning of the data (i.e., partitioning by feature), along with a set of optimized data structures to reduce the CPU and communication costs of training. YGGDRASIL (1) trains directly on compressed data for compressible features and labels; (2) introduces efficient data structures for training on uncompressed data; and (3) minimizes communication between nodes by using sparse bitvectors. Moreover, while PLANET approximates split points through feature binning, YG-GDRASIL does not require binning, and we analytically characterize the impact of this approximation. We evaluate YGGDRASIL against the MNIST 8M dataset and a high-dimensional dataset at Yahoo; for both, YGGDRASIL is faster by up to an order of magnitude.

## 1  Introduction

Decision tree-based methods, such as random forests and gradient-boosted trees, have a rich and successful history in the machine learning literature. They remain some of the most widely-used models for both regression and classification tasks, and have proven to be practically advantageous for several reasons: they are arbitrarily expressive, can naturally handle categorical features, and are robust to a wide range of hyperparameter settings [4].

As datasets have grown in scale, there is an increasing need for distributed algorithms to train decision trees. Google's PLANET framework [12] has been the de facto approach for distributed tree learning, with several popular open source implementations, including Apache Mahout, Spark MLLIB, and XGBOOST [1, 11, 7]. PLANET partitions the training instances across machines and parallelizes the computation of split points and stopping criteria over them, thus effectively leveraging a large cluster.

While PLANET works well for shallow trees and small numbers of features, it has high communication costs when tree depths and data dimensionality grow. PLANET's communication cost is linear in the number of features $p$, and is linear in $2^D$, where $D$ is the tree depth. As demonstrated by several studies [13, 3, 8], datasets have become increasingly high-dimensional (large $p$) and complex, often requiring high-capacity models (e.g., deep trees with large $D$) to achieve good predictive accuracy.

We present YGGDRASIL, a new distributed tree learning system that scales well to high-dimensional data and deep trees. Unlike PLANET, YGGDRASIL is based on vertical partitioning of the data [5]: it assigns a subset of the features to each worker machine, and asks it to compute an optimal split for each of its features. These candidate splits are then sent to a master, which selects the best one. On top of the basic idea of vertical partitioning, YGGDRASIL introduces three novel optimizations:

- **Training on compressed data without decompression**: YGGDRASIL compresses features via run-length encoding and encodes labels using dictionary compression. We design a novel split-finding scheme that trains directly on compressed data for compressible features, which reduces runtime by up to 20%.

- **Efficient training on uncompressed data**: YGGDRASIL's data structures let each worker implicitly store the split history of the tree without introducing any memory overheads. Each worker requires only a sequential scan over the data to perform greedy split-finding across all leaf nodes in the tree, and only one set of sufficient statistics is kept in memory at a time.

- **Minimal communication between nodes**: YGGDRASIL uses sparse bit vectors to reduce inter-machine communication costs during training.

Together, these optimizations yield an algorithm that is asymptotically less expensive than PLANET on high-dimensional data and deep trees: YGGDRASIL's communication cost is $O(2^D + Dn)$, in contrast to $O(2^D p)$ for PLANET-based methods, and its data structure optimizations yield up to $2\times$ savings in memory and 40% savings in time over a naive implementation of vertical partitioning. These optimizations enable YGGDRASIL to scale up to thousands of features and tree depths up to 20. On tree depths greater than 10, YGGDRASIL outperforms MLLIB and XGBOOST by up to $6\times$ on the MNIST 8M dataset, and up to $24\times$ on a dataset with 2 million training examples and 3500 features modeled after the production workload at Yahoo.

**Notation** We define $n$ and $p$ as the number of instances and features in the training set, $D$ as the maximum depth of the tree, $B$ as the number of histogram buckets to use in PLANET, $k$ as the number of workers in the cluster, and $W_j$ as the $j$th worker.

## 2   PLANET: Horizontal Partitioning

We now describe the standard algorithm for training a decision tree in a distributed fashion via horizontal partitioning, inspired by PLANET [12]. We assume that $B$ potential thresholds for each of the $p$ features are considered; thus, we define $\mathcal{S}$ as the set of cardinality $pB$ containing all split candidates. For a given node $i$, define the set $\mathcal{I}$ as the instances belonging to this node. We can then express the splitting criterion $Split(\cdot)$ for node $i$ as $Split(i) = \arg\max_{s \in \mathcal{S}} f(\sum_{\mathbf{x} \in \mathcal{I}} g(\mathbf{x}, s))$ for functions $f$ and $g$ where $f : \mathbb{R}^c \to \mathbb{R}$, $g : \mathbb{R}^p \times \mathbb{N} \to \mathbb{R}^c$, and $c \in O(1)$. Intuitively, for each split candidate $s \in \mathcal{S}$, $g(\mathbf{x}, s)$ computes the $c$ sufficient statistics for each point $\mathbf{x}$; $f(\cdot)$ aggregates these sufficient statistics to compute the node purity for candidate $s$; and $Split(i)$ returns the split candidate that maximizes the node purity. Hence, if worker $j$ contains instances in the set $\mathcal{J}$, we have:

$$Split(i) = \arg\max_{s \in \mathcal{S}} f\left(\sum_{j=1}^{k} g_j(s)\right) \quad \text{where} \quad g_j(s) = \sum_{x \in \mathcal{I} \cap \mathcal{J}} g(\mathbf{x}, s) \tag{1}$$

This observation suggests a natural distributed algorithm. We assume a star-shaped distributed architecture with a master node and $k$ worker nodes. Data are distributed using horizontal partitioning; i.e., each worker node stores a subset of the $n$ instances. For simplicity, we assume that we train our tree to a fixed depth $D$. On the $t$th iteration of the algorithm, we compute the optimal splits for all nodes on the $t$th level of the tree via a single round trip of communication between the master and the workers. Each tree node $i$ is split as follows:

1. The $j$th worker locally computes sufficient statistics $g_j(s)$ from Equation 1 for all $s \in \mathcal{S}$.
2. Each worker communicates all statistics $g_j(s)$ to the master ($Bp$ in total).
3. The master computes the best split $s^* = Split(i)$ from Equation 1.
4. The master broadcasts $s^*$ to the workers, who update their local states to keep track of which instances are assigned to which child nodes.

Overall, the computation is linear in $n$, $p$, and $D$, and is trivially parallelizable. Yet the algorithm is communication-intensive. For each tree node, step 2 above requires communicating $kBp$ tuples of size $c$. With $2^D$ total nodes, the total communication is $2^D kpBc$ floating point values, which is exponential in tree depth $D$, and linear in $B$, the number of thresholds considered for each feature. Moreover, using $B < n$ thresholds results in an approximation of the tree trained on a single machine, and can result in adverse statistical consequences, as noted empirically by [7]. We present a theoretical analysis of the impact of this approximation in Section 4.

# 3 YGGDRASIL: Vertical Partitioning

We propose an alternative algorithm to address the aforementioned shortcomings. Rather than partition the data by instance, we partition by feature: each of the $k$ worker machines stores all feature values for $\lceil \frac{p}{k} \rceil$ of the features, as well the labels for all instances. This organizational strategy has two crucial benefits: (1) each worker can locally compute the node purity for a subset of the split candidates, which significantly reduces the communication bottleneck; and (2) we can efficiently consider all possible $B = n - 1$ splits.

We can derive an expression for $Split(i)$ with vertical partitioning analogous to Equation 1. Redefining $\mathcal{J}$ to be the set of features stored on the $j$th worker, we have

$$Split(i) = \arg\max_{j} f_j \quad \text{where} \quad f_j = \arg\max_{s \in \mathcal{J}} f\left( \sum_{\mathbf{x} \in \mathcal{I}} g(\mathbf{x}, s) \right) \tag{2}$$

Intuitively, each worker identifies its top split candidate among its set of features, and the master then chooses the best split candidate among these top $k$ candidates.

As with horizontal partitioning, computation is linear in $n$, $p$, and $D$, and is easily parallelizable. However, the communication profile is quite different, with two major sources of communication. For each node, each worker communicates one tuple of size $c$, resulting in $2^D kc$ communication for all nodes. When training each level of the tree, $n$ bits are communicated to indicate the split direction (left/right) for each training point. Hence, the overall communication is $O(2^D k + Dnk)$. In contrast to the $O(2^D kpB)$ communication cost of horizontal partitioning, vertical partitioning has no dependence on $p$, and, for large $n$, the $O(Dnk)$ term will likely be the bottleneck.

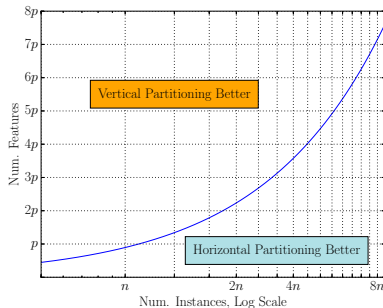

(a) Regimes of $(n, p)$ where each partitioning strategy dominates for $D = 15$, $k = 16$, $B = 32$.

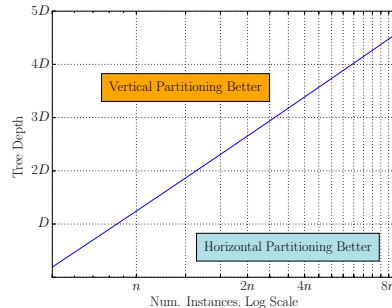

(b) Regimes of $(n, D)$ where each partitioning strategy dominates for $p = 2500$, $k = 16$, $B = 32$.

Figure 1: Communication cost tradeoffs between vertical and horizontal partitioning

Thus, there exists a set of tradeoffs between horizontal and vertical partitioning across different regimes of $n$, $p$, and $D$, as illustrated in Figure 1. The overall trend is clear: for large $p$ and $D$, vertical partitioning can drastically reduce communication.

## 3.1 Algorithm

The YGGDRASIL algorithm works as follows: at iteration $t$, we compute the optimal splits for all nodes on the $t$th level of the tree via two round trips of communication between the master and the workers. Like PLANET, all splits for a single depth $t$ are computed at once. For each node $i$ at depth $t$, the following steps are performed:

`ComputeBestSplit`($i$):
- The $j$th worker locally computes $f_j$ from Equation 2 and sends this to the master.
- The master selects $s^* = Split(i)$. Let $f_j^*$ denote the optimal feature selected for $s^*$, and let $W_j^*$ be the worker containing this optimal feature: $f_j^* \in W_j^*$.

$bitVector =$ `CollectBitVector`($W_j^*$):

- The master requests a bitvector from $W_j^*$ in order to determine which child node (either left or right) each training point $\mathbf{x} \in \mathcal{I}$ should be assigned to.

`BroadcastSplitInfo`($bitVector$):
- The master then broadcasts the bitvector to all $k$ workers. Each worker then updates its internal state to prepare for the next iteration of training.

## 3.2 Optimizations

As we previously showed, vertical partitioning leads to asymptotically lower communication costs as $p$ and $D$ increase. However, this asymptotic behavior does not necessarily translate to more efficient tree learning; on the contrary, a naive implementation may easily lead to high CPU and memory overheads, communication overhead, and poor utilization of the CPU cache. In YGGDRASIL, we introduce three novel optimizations for vertically partitioned tree learning that significantly improve its scalability, memory usage and performance.

### 3.2.1 Sparse Bitvectors for Reduced Communication Overhead

Once the master has found the optimal split $s^*$ for each leaf node $i$ in the tree, each worker must then update its local features to reflect that the instances have been divided into new child nodes. To accomplish this while minimizing communication, the workers and master communicate using bitvectors. Specifically, after finding the optimal split, the master requests from worker $W_j^*$ a corresponding bitvector for $s^*$; this bitvector encodes the partitioning of instances between the two children of $i$. Once the master has collected all optimal splits for all leaf nodes, it broadcasts the bitvectors out to all workers. This means that (assuming a fully balanced tree), for every depth $t$ during training, $2^t$ bitvectors – for a total of $n$ bits – are sent from the $k$ workers.

Additionally, the $n$ bits are encoded in a sparse format [6], which offers much better compression via packed arrays than a naive bitvector. This sparse encoding is particularly useful for imbalanced trees: rather than allocate memory to encode a potential split for all nodes at depth $t$, we only allocate memory for the nodes in which an optimal split was found. By taking advantage of sparsity, we can send the $n$ bits between the master and the workers at only a fraction of the cost.

### 3.2.2 Training on Compressed Data without Decompression

In addition to its more favorable communication cost for large $p$ and $D$, YGGDRASIL's vertical partitioning strategy presents a unique optimization opportunity: the ability to efficiently compress data by feature. Furthermore, because the feature values must be in sorted order to perform greedy split-finding, we can use this to our advantage to perform lossless compression without sacrificing recoverability. This leads to a clear optimization: feature compression via run-length encoding (RLE), an idea that has been explored extensively in column-store databases [10, 14]. In addition to the obvious in-memory savings, this technique also impacts the runtime performance of split-finding, since the vast majority of feature values are now able to reside in the L3 cache. To the best of our knowledge, YGGDRASIL is the first system to apply this optimization to decision tree learning.

Many features compress well using RLE: sparse features, continuous features with few distinct values, and categorical features with low arity. However, to train directly on compressed data without decompressing, we must maintain the feature in sorted order throughout the duration of training, a prerequisite for RLE. Therefore, to compute all splits for a given depth $t$, we introduce a data structure to record the most recent splits at depth $t - 1$. Specifically, we create a mapping between each feature value and the node $i$ at depth $t$ that it is currently assigned to.

At the end of an iteration of training, each worker updates this data structure by applying the bitvector it receives from the master, which requires a single sequential scan over the data. All random accesses are confined to the labels, which we also encode (when feasible) using dictionary compression. This gives us much better cache density during split-finding: all random accesses no longer touch DRAM and instead read from the last-level cache.

To minimize the number of additional passes, we compute the optimal split across *all leaf nodes* as we iterate over a given feature. This means that each feature requires only two sequential scans over the data for each iteration of training: one to update the value-node mapping, and one to compute the entire set of optimal splits for iteration $t + 1$. However, as a tradeoff, we must maintain the

sufficient statistics for all splits in memory as we scan over the feature. For categorical features (especially those with high arity), this cost in memory overhead proves to be too exorbitant, and the runtime performance suffers despite obtaining excellent compression. For sparse continuous features, however, the improvements are significant: on MNIST 8M, we achieve $2\times$ compression (including the auxiliary data structure) and obtain a 20% reduction in runtime.

### 3.2.3 Efficient Training on Uncompressed Data

For features that aren't highly compressible, YGGDRASIL uses a different scheme that, in contrast, does not use any auxiliary data structures to keep track of the split history. Since features no longer need to stay sorted in perpetuity, YGGDRASIL implicitly encodes the split partitions by recursively dividing its features into sub-arrays – each feature value is assigned to a sub-array based on the bit assigned to it and its previous sub-array assignment. Because the feature is initially sorted, a sequential scan over the sub-arrays maintains the sorted-order invariant, and we construct the sub-arrays for the next iteration of training in $O(n)$ time, requiring only a single pass over the feature. By using this implicit representation of the split history, we're left only with the feature values and label indices stored in memory. Therefore, the memory load does not increase during training for uncompressed features – it remains constant at $2\times$.

This scheme yields another additional benefit: when computing the next iteration of splits for depth $t + 1$, YGGDRASIL only maintains the sufficient statistics for one node at a time, rather than for all leaf nodes. Furthermore, YGGDRASIL still only requires a single sequential scan through the entire feature to compute all splits. This means that, as was the case for compressed features, every iteration of training requires only two sequential scans over each feature, and all random accesses are again confined to the dictionary-compressed labels. Finally, for imbalanced trees, we can skip entire sub-arrays that no longer need to be split, which saves additional time as trees grow deeper.

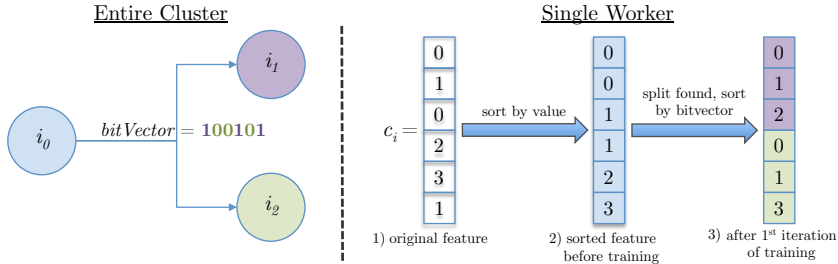

Figure 2: Overview of one iteration of uncompressed training in YGGDRASIL. **Left side**: Root node $i_0$ is split into nodes $i_1$ and $i_2$; the split is encoded by a bitvector. **Right side**: Prior to training, the feature $c_i$ is sorted to optimize split-finding. Once a split has been found, $c_i$ is re-sorted into two sub-arrays: the 1st, 4th, and last values (the "on" bits) are sorted into $i_1$'s sub-array, and the "off" bits are sorted into $i_2$'s sub-array. Each sub-array is in sorted order for the next iteration of training.

## 4 Discretization Error Analysis

Horizontal partitioning requires each worker to communicate the impurity on its subset of data for all candidate splits associated with each of the $p$ features, and to do so for all $2^D$ tree nodes. For continuous features where each training instance could have a distinct value, up to $n$ candidate splits are possible so the communication cost is $O(2^D kpn)$. To improve efficiency, continuous features are instead commonly discretized to $B$ discrete bins such that only $B$ rather than $n - 1$ candidate splits are considered at each tree node [9]. In contrast, discretization of continuous features is not required in YGGDRASIL, since all $n$ values for a particular feature are stored on a single worker (due to vertical partitioning). Hence, the impurity for the best split rather than all splits is communicated.

This discretization heuristic results in the approximation of continuous values by discrete representatives, and can adversely impact the statistical performance of the resulting decision tree, as demonstrated empirically by [7]. Prior work has shown that the number of bins can be chosen such that the decrease in information gain at any internal tree node between the continuous and discretized

feature can be made arbitrarily small [2]. However, their proof does not quantify the effects of discretization on a decision tree's performance.

To understand the impact of discretization on accuracy, we analyze the simplified setting of training a decision stump classifier on a single continuous feature $x \in [0, 1]$. Suppose the feature data is drawn i.i.d. from a uniform distribution, i.e., $x^{(i)} \stackrel{iid}{\sim} \mathcal{U}[0, 1]$ for $i = 1, 2, \ldots, n$, and that labels are generated according to some threshold $t^{truth} \sim \mathcal{U}[0, 1]$, i.e., $y^{(i)} = \text{sgn}(x^{(i)} - t^{\text{truth}})$. The decision stump training criterion seeks to choose a splitting threshold at one of the training instances $x^{(i)}$ in order to minimize $\hat{t}_n = \arg\max_{t \in \{x^{(i)}\}} f(t)$, where $f(t)$ is some purity measure. In our analysis, we will define the purity measure to be information gain. In our simplified setting, we show that there is a natural relationship between the misclassification probability $P_{err}(t)$, and the approximation error of our decision stump, i.e., $|t - t^{truth}|$. All proofs are deferred to the appendix.

**Observation 1.** *For an undiscretized decision stump, as $n \to \infty$, $P_{err}(\hat{t}_n) \stackrel{a.s.}{\to} 0$.*

**Observation 2.** *Maximizing information gain is equivalent to minimizing absolute distance, i.e.,*

$$\hat{t}_n = \arg\max_{t \in \{x^{(i)}\}_{i=1}^n} f(t) = \arg\min_{t \in \{x^{(i)}\}_{i=1}^n} |t - t^{truth}|.$$

*Moreover, $P_{err}(t) = |t - t^{truth}|$.*

We now present our main result. This intuitive result shows that increasing the number of discretization bins $B$ leads to a reduction in the expected probability of error.

**Theorem 1.** *Let $\hat{t}_{N,B}$ denote the threshold learned by a decision stump on $n$ training instances discretized to $B + 1$ levels. Then $E\left[P_{err}(\hat{t}_{N,B})\right] \stackrel{a.s.}{\to} \frac{1}{4B}$.*

## 5  Evaluation

We developed YGGDRASIL on top of Spark 1.6.0 with an API compatible with MLLIB. Our implementation is 1385 lines of code, excluding comments and whitespace. Our implementation is open-source and publicly available.[1] Our experimental results show that, for large $p$ and $D$, YGGDRASIL outperforms PLANET by an order of magnitude, corroborating our analysis in Section 3.

### 5.1  Experimental Setup

We benchmarked YGGDRASIL against two implementations of PLANET: Spark MLLIB v1.6.0, and XGBOOST4J-SPARK v0.47. These two implementations are slightly different from the algorithm from Panda et al. In particular, the original PLANET algorithm has separate subroutines for distributed vs. "local" training. By default, PLANET executes the horizontally partitioned algorithm in Section 2 using on-disk data; however, if the instances assigned to a given tree node fit in-memory on a single worker, then PLANET moves all the data for that node to one worker and switches to in-memory training on that worker. In contrast, MLLIB loads all the data into distributed memory across the cluster at the beginning and executes all training passes in memory. XGBOOST extends PLANET with several additional optimizations; see [7] for details.

We ran all experiments on 16 Amazon EC2 r3.2xlarge machines. Each machine has an Intel Xeon E5-2670 v2 CPU, 61 GB of memory, and 1 Gigabit Ethernet connectivity. Prior to our experiments, we tuned Spark's memory configuration (heap memory used for storage, number of partitions, etc.) for optimal performance. All results are averaged over five trials.

### 5.2  Large-scale experiments

To examine the performance of YGGDRASIL and PLANET, we trained a decision tree on two large-scale datasets: the MNIST 8 million dataset, and another modeled after a private Yahoo dataset that is used for search ranking. Table 1 summarizes the parameters of these datasets.

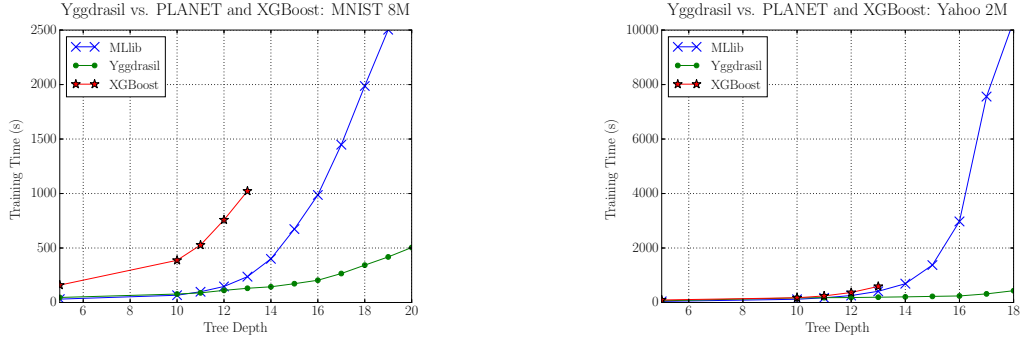

Figure 3: Training time vs. tree depth for MNIST 8M and Yahoo 2M.

| Dataset | # instances | # features | Size | Task |
|---------|-------------|------------|------|------|
| MNIST 8M | $8.1 \times 10^6$ | 784 | 18.2 GiB | classification |
| Yahoo 2M | $2 \times 10^6$ | 3500 | 52.2 GiB | regression |

Table 1: Parameters of the datasets for our experiments

Figure 3 shows the training time across various tree depths for MNIST 8M and Yahoo 2M. For both datasets, we carefully tuned XGBOOST to run on the maximum number of threads and the optimal number of partitions. Despite this, XGBOOST was unable to train trees deeper than $D = 13$ without crashing due to OutOfMemory exceptions. While Spark MLLIB's implementation of PLANET is marginally faster for shallow trees, its runtime increases exponentially as $D$ increases. YGGDRASIL, on the other hand, scales well up to $D = 20$, for which it runs up to $6\times$ faster. For the Yahoo dataset, YGGDRASIL's speed-up is even greater because of the higher number of features $p$ – recall that the communication cost for PLANET is proportional to $2^D$ and $p$. Thus, for $D = 18$, YGGDRASIL is up to $24\times$ faster than Spark MLLIB.

## 5.3 Study of Individual Optimizations

To understand the impacts of the optimizations in Section 3.2, we measure each optimization's effect on YGGDRASIL runtime. To fully evaluate our optimizations – including feature compression – we chose MNIST 8M, whose features are all sparse, for this study. The results are in Figure 6: we see that the total improvement from the naive baseline to the fully optimized algorithm is a 40% reduction in runtime. Using sparse bitvectors reduces the communication overhead between the master and the workers, giving a modest speedup. Encoding the labels and compressing the features via run-length encoding each yield 20% improvements. As discussed, these speedups are due to improved cache utilization: encoding the labels via dictionary compression reduces their size in memory by $8\times$; as a result, the labels entirely fit in the last-level cache. The feature values also fit in cache after applying RLE, and we gain $2\times$ in memory overhead once we factor in needed auxiliary data structures.

## 5.4 Scalability experiments

To further demonstrate the scalability of YGGDRASIL vs. PLANET for high-dimensional datasets, we measured the training time on a series of synthetic datasets parameterized by $p$. For each dataset, approximately $\frac{p}{2}$ features were categorical, while the remaining features were continuous. From Figure 4, we see that, YGGDRASIL scales much more effectively as $p$ increases, especially for larger $D$. In particular, for $D = 15$, YGGDRASIL is initially $3\times$ faster than PLANET for $p = 500$, but is more than $8\times$ faster for $p = 4000$. This confirms our asymptotic analysis in Section 3.

## 6  Related Work

**Vertical Partitioning.**  Several authors have proposed partitioning data by feature for training decision trees; to our knowledge, none of these systems perform the communication and data structure optimizations in YGGDRASIL, and none report results at the same scale. Svore and Burges

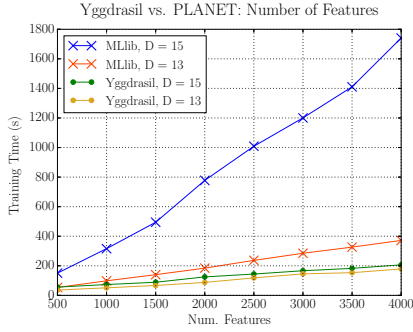

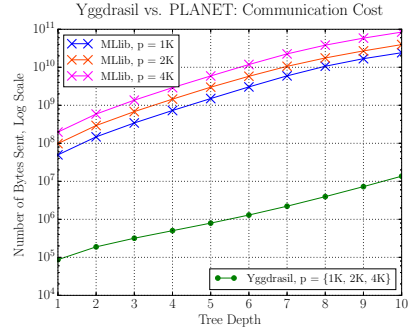

Figure 4: Training time vs. number of features for $n = 2 \times 10^6$, $k = 16$, $B = 32$. Because the communication cost of PLANET scales linearly with $p$, the total runtime increases at a much faster rate.

Figure 5: Number of bytes sent vs. tree depth for $n = 2 \times 10^6$, $k = 16$, $B = 32$. For YGGDRASIL, the communication cost is the same for all $p$; each worker sends its best local feature to the master.

[15] treat data as vertical columns, but place a full copy of the dataset on every worker node, an approach that is not scalable for large datasets. Caragea et al. [5] analyze the costs of horizontal and vertical partitioning but do not include an implementation. Ye et al. [16] implement vertical partitioning using MapReduce or MPI and benchmark data sizes up to 1.2 million rows and 520 features. None of these systems compress columnar data on each node, communicate using sparse bitvectors, or optimize for cache locality as YGGDRASIL does (Section 3.2). These optimizations yield significant speedups over a basic implementation of vertical partitioning (Section 5.3).

**Distributed Tree Learning.** The most widely used distributed tree learning method is PLANET ([12]), which is also implemented in open-source libraries such as Apache Mahout ([1]) and MLLIB ([11]). As shown in Figure 1, PLANET works well for shallow trees and small numbers of features, but its cost grows quickly with tree depth and is proportional to the number of features and the number of bins used for discretization. This makes it suboptimal for some large-scale tree learning problems.

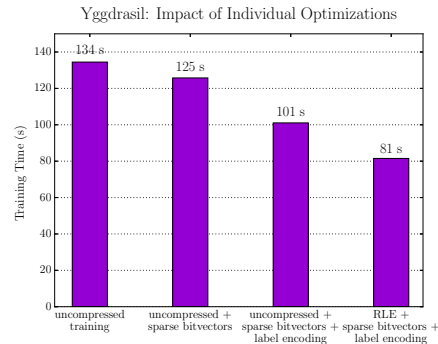

Figure 6: YGGDRASIL runtime improvements from specific optimizations, on MNIST 8M at $D = 10$.

XGBOOST ([7]) uses a partitioning scheme similar to PLANET, but uses a compressed, sorted columnar format inside each "block" of data. Its communication cost is therefore similar to PLANET, but its memory consumption is smaller. XGBOOST is optimized for gradient-boosted trees, in which case each tree is relatively shallow. It does not perform as well as YGGDRASIL on deeper trees, such as those needed for random forests, as shown in our evaluation. XGBOOST also lacks some of the processing optimizations in YGGDRASIL, such as label encoding to maximize cache density and training directly on run-length encoded features without decompressing.

## 7 Conclusion

Decision trees and tree ensembles are an important class of models, but previous distributed training algorithms were optimized for small numbers of features and shallow trees. We have presented YGGDRASIL, a new distributed tree learning system optimized for deep trees and thousands of features. Through vertical partitioning of the data and a set of data structure and algorithmic optimizations, YGGDRASIL outperforms existing tree learning systems by up to 24×, while simultaneously eliminating the need to approximate data through binning. YGGDRASIL is easily implementable on parallel engines like MapReduce and Spark.

## Footnotes

[1]Yggdrasil has been published as a Spark package at the following URL: `https://spark-packages.org/package/fabuzaid21/yggdrasil`

# References

[1] Apache Mahout. `https://mahout.apache.org/`, 2015.

[2] Y. Ben-Haim and E. Tom-Tov. A streaming parallel decision tree algorithm. *The Journal of Machine Learning Research*, 11:849–872, 2010.

[3] L. Breiman. Random forests. *Machine learning*, 45(1):5–32, 2001.

[4] L. Breiman, J. Friedman, C. J. Stone, and R. A. Olshen. *Classification and regression trees*. CRC press, 1984.

[5] D. Caragea, A. Silvescu, and V. Honavar. A framework for learning from distributed data using sufficient statistics and its application to learning decision trees. *International Journal of Hybrid Intelligent Systems*, 1(1, 2):80–89, 2004.

[6] S. Chambi, D. Lemire, O. Kaser, and R. Godin. Better bitmap performance with roaring bitmaps. *Software: Practice and Experience*, 2015.

[7] T. Chen and C. Guestrin. Xgboost: A scalable tree boosting system. *arXiv preprint arXiv:1603.02754*, 2016.

[8] C. Cortes, M. Mohri, and U. Syed. Deep boosting. In *ICML*, 2014.

[9] U. M. Fayyad and K. B. Irani. On the handling of continuous-valued attributes in decision tree generation. *Mach. Learn.*, 8(1):87–102, Jan. 1992. ISSN 0885-6125. doi: 10.1023/A: 1022638503176. URL `http://dx.doi.org/10.1023/A:1022638503176`.

[10] A. Lamb, M. Fuller, R. Varadarajan, N. Tran, B. Vandiver, L. Doshi, and C. Bear. The vertica analytic database: C-store 7 years later. *Proceedings of the VLDB Endowment*, 5(12):1790–1801, 2012.

[11] X. Meng, J. K. Bradley, B. Yavuz, E. R. Sparks, S. Venkataraman, D. Liu, J. Freeman, D. B. Tsai, M. Amde, S. Owen, D. Xin, R. Xin, M. J. Franklin, R. Zadeh, M. Zaharia, and A. Talwalkar. MLlib: Machine learning in apache spark. *arXiv:1505.06807*, 2015.

[12] B. Panda, J. S. Herbach, S. Basu, and R. J. Bayardo. Planet: Massively parallel learning of tree ensembles with mapreduce. *International Conference on Very Large Data Bases*, 2009.

[13] S. R. Safavian and D. Landgrebe. A survey of decision tree classifier methodology. *IEEE transactions on systems, man, and cybernetics*, 21(3):660–674, 1991.

[14] M. Stonebraker, D. J. Abadi, A. Batkin, X. Chen, M. Cherniack, M. Ferreira, E. Lau, A. Lin, S. Madden, E. O'Neil, et al. C-store: a column-oriented dbms. In *Proceedings of the 31st international conference on Very large data bases*, pages 553–564. VLDB Endowment, 2005.

[15] K. M. Svore and C. Burges. Large-scale learning to rank using boosted decision trees. *Scaling Up Machine Learning: Parallel and Distributed Approaches*, 2, 2011.

[16] J. Ye, J.-H. Chow, J. Chen, and Z. Zheng. Stochastic gradient boosted distributed decision trees. In *Proceedings of the 18th ACM Conference on Information and Knowledge Management*, CIKM '09, pages 2061–2064, New York, NY, USA, 2009. ACM. ISBN 978-1-60558-512-3. doi: 10.1145/1645953.1646301. URL `http://doi.acm.org/10.1145/1645953.1646301`.

